# Evaluating multi-class learning strategies in a hierarchical framework for object detection

**Sanja Fidler**    **Marko Boben**    **Aleš Leonardis**
Faculty of Computer and Information Science
University of Ljubljana, Slovenia
{sanja.fidler, marko.boben, ales.leonardis}@fri.uni-lj.si

## Abstract

Multi-class object learning and detection is a challenging problem due to the large number of object classes and their high visual variability. Specialized detectors usually excel in performance, while joint representations optimize sharing and reduce inference time — but are complex to train. Conveniently, sequential class learning cuts down training time by transferring existing knowledge to novel classes, but cannot fully exploit the shareability of features among object classes and might depend on ordering of classes during learning. In hierarchical frameworks these issues have been little explored. In this paper, we provide a rigorous experimental analysis of various multiple object class learning strategies within a generative hierarchical framework. Specifically, we propose, evaluate and compare three important types of multi-class learning: 1.) independent training of individual categories, 2.) joint training of classes, and 3.) sequential learning of classes. We explore and compare their computational behavior (space and time) and detection performance as a function of the number of learned object classes on several recognition datasets. We show that sequential training achieves the best trade-off between inference and training times at a comparable detection performance and could thus be used to learn the classes on a larger scale.

## 1   Introduction

Object class detection has been one of the mainstream research areas in computer vision. In recent years we have seen a significant trend towards larger recognition datasets with an increasing number of object classes [1]. This necessitates representing, learning and detecting multiple object classes, which is a challenging problem due to the large number and the high visual variability of objects.

To learn and represent multiple object classes there have mainly been two strategies: the detectors for each class have either been trained in isolation, or trained on all classes simultaneously. Both exert certain advantages and disadvantages. Training independently allows us to apply complex probabilistic models that use a significant amount of class specific features and allows us to tune the parameters for each class separately. For object class detection, these approaches had notable success [2]. However, representing multiple classes in this way, means stacking together specific class representations. This, on the one hand, implies that each novel class can be added in constant time, however, the representation grows clearly linearly with the number of classes and is thus also linear in inference. On the other hand, joint representations enlarge sublinearly by virtue of sharing the features among several object classes [3, 4]. This means sharing common computations and increasing the speed of the joint detector. Training, however, is usually quadratic in the number of classes. Furthermore, adding just one more class forces us to re-train the representation altogether.

Receiving somewhat less attention, the strategy to learn the classes sequentially (but not independently) potentially enjoys the traits of both learning types [4, 5, 6]. By learning one class after

another, we can transfer the knowledge acquired so far to novel classes and thus likely achieve both, sublinearity in inference and cut down training time. In order to scale to a higher number of object classes, learning them sequentially lends itself as the best choice.

In literature, the approaches have mainly used one of these three learning strategies in isolation. To the best of our knowledge, little research has been done on analyzing and comparing them with respect to one another. This is important because it allows us to point to losses and gains of each particular learning setting, which could focus further research and improve the performance. This is exactly what this paper is set to do — we present a hierarchical framework within which **all** of the aforementioned learning strategies can be unbiasedly evaluated and put into perspective.

Prominent work on these issues has been done in the domain of flat representations [4, 3], where each class is modeled as an immediate aggregate of local features. However, there is an increasing literature consensus, that hierarchies provide a more suitable form of multi-class representation [7, 8, 9, 10, 11, 12]. Hierarchies not only share complex object parts among similar classes, but can re-use features at several levels of granularity also for dissimilar objects.

In this paper, we provide a rigorous experimental evaluation of several important multi-class learning strategies for object detection within a *generative hierarchical framework*. We make use of the hierarchical learning approach by [13]. Here we propose and evaluate three types of multi-class learning: 1.) independent training of individual categories, 2.) joint training, 3.) sequential training of classes. Several issues were evaluated on multiple object classes: 1.) growth of representation, 2.) training and 3.) inference time, 4.) degree of feature sharing and re-use at each level of the hierarchy, 5.) influence of class ordering in sequential learning, and 6.) detection performance, all as a function of the number of classes learned. We show that sequential training achieves the best trade-off between inference and training times at a comparable detection performance and could thus be used to learn the classes on a larger scale.

**Related work.** Prior work on multi-class learning in generative hierarchies either learns separate hierarchies for each class [14, 15, 16, 10, 17], trains jointly [7, 18, 9, 19, 20, 11], whereas work on sequential learning of classes has been particularly scarce [6, 13]. However, to the best of our knowledge, no work has dealt with, evaluated and compared multiple important learning concepts under one *hierarchical* framework.

## 2 The hierarchical model and inference

**The hierarchical model**. We use the hierarchical model of [13, 21], which we summarize here. Objects are represented with a recursive compositional shape vocabulary which is *learned* from images. The vocabulary contains a set of *shape models* or *compositions* at each layer. Each shape model in the hierarchy is modeled as a conjunction of a small number of *parts* (shapes from the previous layer). Each part is spatially constrained on the parent shape model via a spatial relation which is modeled with a two-dimensional Gaussian distribution. The number and the type of parts can differ across the shape models and is learned from the data without supervision. At the lowest layer, the vocabulary consists of a small number of short oriented contour fragments, while the vocabulary at the top-most layer contains models that code the shapes of the whole objects. For training, we need a positive and a validation set of class images, while the structure of the representation is learned in an unsupervised way (no labels on object parts or smaller subparts need to be given).

The hierarchical vocabulary $\mathcal{V} = (V, E)$ is represented with a directed graph, where multiple edges between two vertices are allowed. The vertices $V$ of the graph represent the shape models and the edges $E$ represent the composition relations between them. The graph $\mathcal{V}$ has a hierarchical structure, where the set of vertices $V$ is partitioned into subsets $V^1, \ldots, V^{\mathcal{O}}$, each containing the shapes at a particular layer. The vertices $\{v_i^1\}_{i=1}^6$ at the lowest layer $V^1$ represent 6 oriented contour fragments. The vertices at the top-most layer $V^{\mathcal{O}}$, referred to as the *object layer* represent the whole shapes of the objects. Each object class $C$ is assigned a subset of vertices $V_C^{\mathcal{O}} \subseteq V^{\mathcal{O}}$ that code the object layer shapes of that particular class. We denote the set of edges between the vertex layers $V^\ell$ and $V^{\ell-1}$ with $E^\ell$. Each edge $e_{Ri}^\ell = v_R^\ell v_i^{\ell-1}$ in $E^\ell$ is associated with the Gaussian parameters $\theta_{Ri}^\ell := \theta(e_{Ri}^\ell) = (\mu_{Ri}^\ell, \Sigma_{Ri}^\ell)$ of the spatial relation between the parent shape $v_R^\ell$ and its part $v_i^{\ell-1}$. We will use $\boldsymbol{\theta}_R^\ell = (\theta_{Ri}^\ell)_i$ to denote the vector of all the parameters of a shape model $v_R^\ell$. The pair $\mathcal{V}^\ell := (V^\ell, E^\ell)$ will be referred to as the *vocabulary at layer* $\ell$.

**Inference.** We infer object class instances in a query image $I$ in the following way. We follow the contour extraction of [13], which finds local maxima in oriented Gabor energy. This gives us the contour fragments $\mathbf{F}$ and their positions $\mathbf{X}$. In the process of inference we build a (directed acyclic) *inference graph* $\mathcal{G} = (Z, Q)$. The vertices $Z$ are partitioned into vertex layers 1 to $\mathcal{O}$ (object layer), $Z = Z^1 \cup \cdots \cup Z^{\mathcal{O}}$, and similarly also the edges, $Q = Q^1 \cup \cdots \cup Q^{\mathcal{O}}$. Each vertex $z^\ell = (v^\ell, x^\ell) \in Z^\ell$ represents a *hypothesis* that a particular shape $v^\ell \in V^\ell$ from the vocabulary is present at location $x^\ell$. The edges in $Q^\ell$ connect each parent hypothesis $z_R^\ell$ to all of its part hypotheses $z_i^{\ell-1}$. The edges in the bottom layer $Q^1$ connect the hypotheses in the first layer $Z^1$ with the observations. With $\mathcal{S}(z)$ we denote the subgraph of $\mathcal{G}$ that contains the vertices and edges of all descendants of vertex $z$.

Since our definition of each vocabulary shape model assumes that its parts are conditionally independent, we can calculate the likelihood of the part hypotheses $z_i^{\ell-1} = (v_i^{\ell-1}, x_i^{\ell-1})$ under a parent hypothesis $z_R^\ell = (v_R^\ell, x_R^\ell)$ by taking a product over the individual likelihoods of the parts:

$$p(\mathbf{v}^{\ell-1}, \mathbf{x}^{\ell-1} \mid v_R^\ell, x_R^\ell, \boldsymbol{\theta}_R^\ell) = \prod_{e_{Ri}^\ell = v_R^\ell v_i^{\ell-1}} p(x_i^{\ell-1} \mid x_R^\ell, v_i^{\ell-1}, v_R^\ell, \theta_{Ri}^\ell) \tag{1}$$

The term $p_{Ri} := p(x_i^{\ell-1} \mid x_R^\ell, v_i^{\ell-1}, v_R^\ell, \theta_{Ri}^\ell)$ stands for the spatial constraint imposed by a vocabulary edge $e_{Ri}^\ell$ between a parent hypothesis $z_R^\ell$ and its part hypothesis $z_i^{\ell-1}$. It is modeled by a normal distribution, $p_{Ri} = \mathcal{N}(x_i^{\ell-1} - x_R^\ell \mid \theta_{Ri}^\ell)$, where $\theta_{Ri}^\ell = (\mu_{Ri}^\ell, \Sigma_{Ri}^\ell)$. If the likelihood in (1) is above a threshold, we add edges between $z_R^\ell$ and its most likely part hypotheses. The log-likelihood of the observations under a hypothesis $z_R^\ell$ is then calculated recursively over the subgraph $\mathcal{S}(z_R^\ell)$:

$$\log p(\mathbf{F}, \mathbf{X}, \mathbf{z}^{1:\ell-1} \mid z_R^\ell; \mathcal{V}) = \sum_{z_{R'} z_{i'} \in E(\mathcal{S}(z_R^\ell))} \log p_{R'i'} + \sum_{z_{i'}^1 \in V(\mathcal{S}(z_R^\ell))} \log p(\mathbf{F}, \mathbf{X} \mid z_{i'}^1), \tag{2}$$

where $E(\mathcal{S}(z_R^\ell))$ and $V(\mathcal{S}(z_R^\ell))$ denote the edges and vertices of the subgraph $S(z_R^\ell)$, respectively. The last term is the likelihood of the Gabor features under a particular contour fragment hypothesis.

## 3 Multi-class learning strategies

We first define the objective function for multi-class learning and show how different learning strategies can be used with it in the following subsections. Our goal is to find a hierarchical vocabulary $\mathcal{V}$ that well represents the distribution $p(I \mid C) \approx p(\mathbf{F}, \mathbf{X} \mid C; \mathcal{V})$ at minimal complexity of the representation ($C$ denotes the class variable). Specifically, we seek for a vocabulary $\mathcal{V} = \cup_\ell \mathcal{V}^\ell$ that optimizes the function $f$ over the data $D = \{(\mathbf{F}_n, \mathbf{X}_n, C_n)\}_{n=1}^N$ ($N$ training images):

$$\mathcal{V}^* = \arg\max_{\mathcal{V}} f(\mathcal{V}), \quad \text{where} \quad f(\mathcal{V}) = L(D \mid \mathcal{V}) - \lambda \cdot T(\mathcal{V}) \tag{3}$$

The first term in (3) represents the log-likelihood:

$$L(D \mid \mathcal{V}) = \sum_{n=1}^N \log p(\mathbf{F}_n, \mathbf{X}_n \mid C; \mathcal{V}) = \sum_{n=1}^N \log \sum_{\mathbf{z}} p(\mathbf{F}_n, \mathbf{X}_n, \mathbf{z} \mid C; \mathcal{V}), \tag{4}$$

while $T(\mathcal{V})$ penalizes the complexity of the model [21] and $\lambda$ controls the amount of penalization.

Several approximations are made to learn the vocabulary; namely, the vocabulary is learned layer by layer (in a bottom-up way) by finding frequent spatial layouts of parts from the previous layer [13] and then using $f$ to select a minimal set of models at each layer that still produce a good whole-object shape representation at the final, object layer [21]. The top layer models are validated on a set of validation images and those yielding a high rate of false-positives are removed from $\mathcal{V}$.

We next show how different training strategies are performed to learn a joint multi-class vocabulary.

### 3.1 Independent training of individual classes

In independent training, a class specific vocabulary $\mathcal{V}_c$ is learned using the training images of each particular class $C = c$. We learn $\mathcal{V}_c$ by maximizing $f$ over the data $D = \{(\mathbf{F}_n, \mathbf{X}_n, C = c)\}$. For the negative images in the validation step, we randomly sample images from other classes. The joint multi-class representation $\mathcal{V}$ is then obtained by stacking the class specific vocabularies $\mathcal{V}_c$ together, $V^\ell = \cup_c V_c^\ell$ (the edges $E$ are added accordingly). Note that $V_c^1$ is the only layer common to all classes (6 oriented contour fragments), thus $V^1 = V_c^1$.

## 3.2 Joint training of classes

In joint training, the learning phase has two steps. In the first step, the training data $D$ for all the classes is presented to the algorithm simultaneously, and is treated as unlabeled. The spatial parameters $\theta$ of the models at each layer are then inferred from images of all classes, and will code "average" spatial part dispositions. The joint statistics also influences the structure of the models by preferring those that are most repeatable over the classes. This way, the jointly learned vocabulary $\mathcal{V}$ will be the best trade-off between the likelihood $L$ and the complexity $T$ over *all* the classes in the dataset. However, the final, top-level likelihood for each particular class could be low because the more discriminative class-specific information has been lost. Thus, we employ a second step which revisits each class separately. Here, we use the joint vocabulary $\mathcal{V}$ and *add* new models $v_R^\ell$ to each layer $\ell$ if they further increase the score $f$ for each particular class. This procedure is similar to that used in sequential training and will be explained in more detail in the following subsection. Object layer $\mathcal{V}^{\mathcal{O}}$ is consequently learned and added to $\mathcal{V}$ for each class. We validate the object models after all classes have been trained. A similarity measure is used to compare every two classes based on the degree of feature sharing between them. In validation, we choose the negative images by sampling the images of the classes according to the distribution defined by the similarity measure. This way, we discard the models that poorly discriminate between the similar classes.

## 3.3 Sequential training of classes

When training the classes sequentially, we train on each class separately, however, our aim is to 1.) maximize the re-use of compositions learned for the previous classes, and 2.) add those missing (class-specific) compositions that are needed to represent class $k$ sufficiently well. Let $\mathcal{V}_{1:k-1}$ denote the vocabulary learned for classes 1 to $k-1$. To learn a novel class $k$, for each layer $\ell$ we seek a new set of shape models that maximizes $f$ over the data $D = \{(\mathbf{F}_n, \mathbf{X}_n, C = k)\}$ conditionally on the already learned vocabulary $\mathcal{V}_{1:k-1}^\ell$. This is done by treating the hypotheses inferred with respect to $\mathcal{V}_{1:k-1}^\ell$ as fixed, which gives us a starting value of the score function $f$. Each new model $v_R^\ell$ is then evaluated and selected conditionally on this value, *i.e* such that the difference $f(\mathcal{V}_{1:k-1}^\ell \cup v_R^\ell) - f(\mathcal{V}_{1:k-1}^\ell)$ is maximized. Since according to the definition in (4) the likelihood $L$ increases the most when the hypotheses have largely disjoint supports, we can greatly speed up the learning process: the models need to be learned only with respect to those $(\mathbf{F}, \mathbf{X})$ in an image that have a low likelihood under the vocabulary $\mathcal{V}_{1:k-1}^\ell$, which can be determined prior to training.

## 4 Experimental results

We have evaluated the hierarchical multi-class learning strategies on several object classes. Specifically, we used: UIUC multi-scale cars [22], GRAZ [4] cows and persons, Weizmann multi-scale horses (adapted by Shotton et al. [23]), all five classes from the ETH dataset [24], and all ten classes from TUD $shape2$ [25]. Basic information is given in Table 1. A 6-layer vocabulary was learned. [1] The bounding box information was used during training.

When evaluating detection performance, a detection will be counted as correct, if the predicted bounding box coincides with groundtruth more than $50\%$. On the ETH dataset alone, this threshold is lowered to 0.3 to enable a fair comparison with the related work [24]. The performance will be given either with recall at equal error rate (EER), positive detection rate at low FPPI, or as classif.-by-detection (on TUD $shape2$), depending on the type of results reported on that dataset thus-far.

To evaluate the shareability of compositions between the classes, we will use the following measure:

$$\text{deg\_share}(\ell) = \frac{1}{|V^\ell|} \sum_{v_R^\ell \in V^\ell} \frac{(\# \text{ of classes that use } v_R^\ell) - 1}{\# \text{ of all classes} - 1},$$

defined for each layer $\ell$ separately. By "$v_R^\ell$ used by class $C$" it is meant that there is a path of edges connecting any of the class specific shapes $V_C^{\mathcal{O}}$ and $v_R^\ell$. To give some intuition behind the measure:

$deg\_share = 0$ if no shape from layer $\ell$ is shared (each class uses its own set of shapes), and it is $1$ if each shape is used by all the classes. Beside the mean (which defines $deg\_share$), the plots will also show the standard deviation. In sequential training, we can additionally evaluate the degree of re-use when learning each novel class. Higher re-use means lower training time and a more compact representation. We expect a tendency of higher re-use as the number $k$ of classes grows, thus we define it with respect to the number of learned classes:

$$\text{deg\_transfer}(k, \ell) = \frac{\text{\# of } v_R^\ell \in V_{1:k-1}^\ell \text{ used by } c_k}{\text{\# of all } v_R^\ell \in V_{1:k}^\ell \text{ used by } c_k} \tag{5}$$

Evaluation was performed by progressively increasing the number of object classes (from 2 to 10). The individual training will be denoted by $I$, joint by $J$, and sequential by $S$.

Table 2 relates the detection performances of $I$ to those of the related work. On the left side, we report detection accuracy at low FPPI rate for the ETH dataset, averaged over 5 random splits of training/test images as in [24]. On the right side, recall at EER is given for a number of classes.

**Two classes.** We performed evaluation on two visually very similar classes (cow, horse), and two dissimilar classes (person, car). Table 3 gives information on 1.) size (the number of compositions at each layer), 2.) training and 3.) inference times, 4.) recall at EER. In sequential training, both possible orders were used (denoted with $S1$ and $S2$) to see whether different learning orders (of classes) affect the performance. The first two rows show the results for each class individually, while the last row contains information with respect to the conjoined representations. Already for two classes, the cumulative training time is slightly lower for $S$ than $I$, while both being much smaller than that of $J$.

*Degree of sharing.* The hierarchies learned in $I$, $J$, and $S$ on cows and horses, and $J$ for car-person are shown in Fig. 2 in a respective order from left to right. The red nodes depict cow/car and blue horse/person compositions. The green nodes depict the shared compositions. We can observe a slightly lower number of shareable nodes for $S$ compared to $J$, yet still the lower layers for cow-horse are almost completely re-used. Even for the visually dissimilar classes (car-person) sharing is present at lower layers. Numerically, the degrees of sharing and transfer are plotted in Fig. 1.

*Detection rate.* The recall values for each class are reported in Table 3. Interestingly, "knowing" horses improved the performance for cows. For car-person, individual training produced the best result, while training person before car turned out to be a better strategy for $S$. Fig. 1 shows the detection rates for cows and horses on the **joint** test set (the strongest class hypothesis is evaluated), which allows for a much higher false-positive rate. We evaluate it with F-measure (to account for FP). A higher performance for all joint representations over the independent one can be observed. This is due to the high degree of sharing in $J$ and $S$, which puts similar hypotheses in perspective and thus discriminates between them better.

**Five classes.** The results for ETH-5 are reported in Table 4. We used half of the images for training, and the other half for testing. The split was random, but the same for $I$, $J$, and $S$. We also test whether different orders in $S$ affect performance (we report an average over 3 random $S$ runs). Ordering does slightly affect performance, which means we may try to find an optimal order of classes in training. We can also observe that the number of compositions at each layer is higher for $S$ as for $J$ (both being much smaller than $I$), but this only slightly showed in inference times.

**Ten classes.** The results on TUD-10 are presented in Table 5. A few examples of the learned shapes for $S$ are shown in Fig. 3. Due to the high training complexity of $J$, we have only ran $J$ for 2, 5 and 10 classes. We report classif.-by-detection (the strongest class hypothesis in an image must overlap with groundtruth more than $50\%$). To demonstrate the strength of our representation, we have also ran (linear) SVM on top of hypotheses from Layers $1 - 3$, and compared the performances. Already here, Layer 3 + SVM outperforms prior work [25] by $10\%$. Fig. 4-(11.) shows classification as a number of learned classes. Our approach consistently outperforms SVM, which is likely due to the high scale- and rotation- variability of images with which our approach copes well. Fig. 4 shows: inference time, cumulative training time, degree of sharing (for the final 10-class repr.), transfer, and classification rates as a function of the number of learned classes.

*Vocabulary size.* The top row in Fig 4 shows representation size for $I$, $J$ and $S$ as a function of learned classes. With respect to worst case ($I$), both $J$ and $S$ have a highly sublinear growth. Moreover, in layers 2 and 3, where the burden on inference is the highest (the highest number of

inferred hypotheses), an almost constant tendency can be seen. We also compare the curves with those reported for a flat approach by Opelt et al. [4] in Fig 4-(5). We plot the number of models at Layer 5 which are approximately of the same granularity as the learned boundary parts in [4]. Both, $J$ and $S$ hierarchical learning types show a significantly better logarithmic tendency as in [4].

Fig 4-(6) shows the size of the hierarchy file stored on disk. It is worth emphasizing that the hierarchy subsuming 10 classes uses only 0.5Mb on disk and could fit on an average mobile device.

**50 classes.** To increase the scale of the experiments we show the performance of sequential training on 50 classes from LabelMe [1]. The results are presented in Fig. 5. For $I$ in the inference time plot we used the inference time for the first class linearly extrapolated with the number of classes. We can observe that $S$ achieves much lower inference times than $I$, although it is clear that for a higher number of classes more research is needed to cut down the inference times to a practical value.

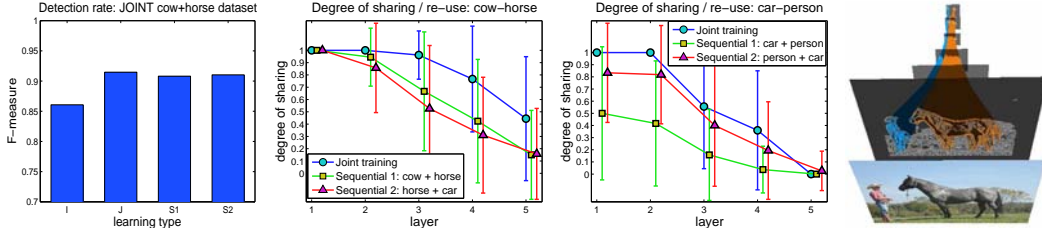

Figure 1: From left to right: 1.) detection rate (F measure) on the joint cow-horse test set. 2.) *degree of sharing* for cow-horse, 3.) car-person vocabularies, 4.) an example detection of a person and horse.

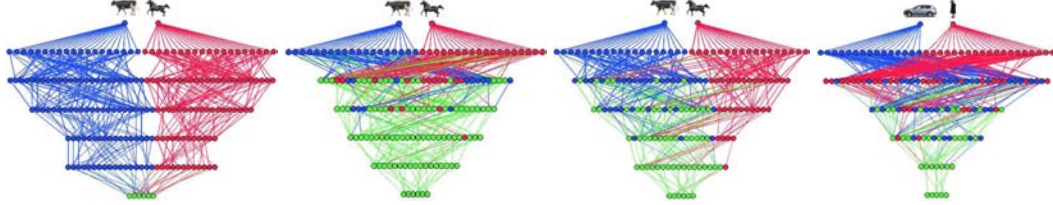

Figure 2: Learned 2-class vocabularies for different learning types (the nodes depict the compositions $v_R^\ell$, the links represent the edges $e_{Ri}^\ell$ between them — the parameters $\boldsymbol{\theta}^\ell$ are not shown). From left to right: cow-horse hierarchy for 1.) $I$, 2.) $J$, 3.) $S1$, and 4.) car-person $J$. Green nodes denoted shared compositions.

# 5 Conclusions and discussion

We evaluated three types of multi-class learning strategies in a hierarchical compositional framework, namely 1.) independent, 2). joint, and 3.) sequential training. A comparison was made through several important computational aspects as well as by detection performance. We conclude that: 1.) Both joint and sequential training strategies exert sublinear growth in vocabulary size (more evidently so in the lower layers) and, consequently, sublinear inference time. This is due to a high degree of sharing and transfer within the resulting vocabularies. The hierarchy obtained by sequential training grows somewhat faster, but not significantly so. 2.) Training time was expectedly worst for joint training, while training time even reduced with each additional class during sequential training. 3.) Different training orders of classes did perform somewhat differently — this means we might try to find an "optimal" order of learning. 4.) Training independently has mostly yielded the best detection rates, but the discrepancy with the other two strategies was low. For similar classes (cow-horse), sequential learning even improved the detection performance, and was in most cases above the joint's performance. By training sequentially, we can learn class specific features (yet still have a high degree of sharing) which boost performance. Most importantly, sequential training has achieved the best trade-off between detection performance, re-usability, inference and training time. The observed computational properties of all the strategies in general, and sequential learning in particular, go well beyond the reported behavior of flat approaches [4]. This makes sequential learning of compositional hierarchies suitable for representing the classes on a larger scale.

**Acknowledgments**

This research has been supported in part by the following funds: EU FP7-215843 project POETICON, EU FP7-215181 project CogX, Research program Computer Vision P2-0214 and Project J2-2221 (ARRS).

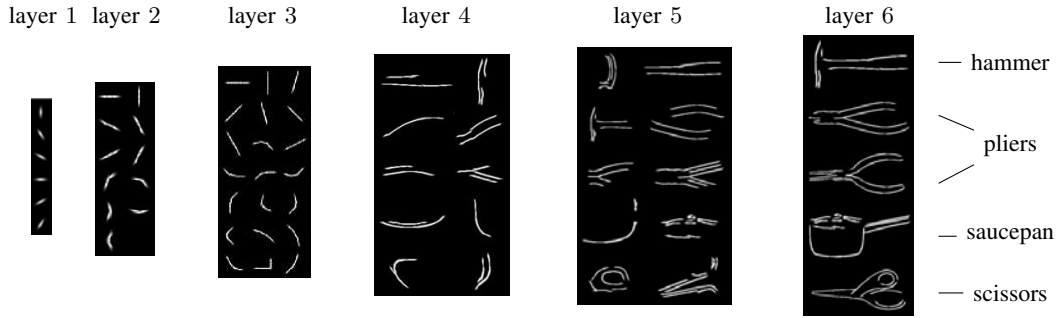

Figure 3: A few examples from the learned hierarchical shape vocabulary for $S$ on TUD-10. Each shape in the hierarchy is a composition of shapes from the layer below. Only the *mean* of each shape is shown.

| method | size on disk | classf. rate | train. time | infer. time | size of representation | | | |
|---|---|---|---|---|---|---|---|---|
| | | | | | L2 | L3 | L4 | L5 |
| Stark et al.[25] | / | 44% | | | | | | |
| Level 1 + SVM | 206 Kb | 32% | | | | | | |
| Level 2 + SVM | 3,913 Kb | 44% | | | | | | |
| Level 3 + SVM | 34,508 Kb | 54% | | | | | | |
| Independent | 1,249 Kb | 71% | 207 min | 12.2 sec | 74 | 96 | 159 | 181 |
| Joint | 408 Kb | 69% | 752 min | 2.0 sec | 14 | 23 | 39 | 59 |
| Sequential | 490 Kb | 71% | 151 min | 2.4 sec | 9 | 21 | 49 | 76 |

Table 5: Results on the TUD-10. Classification obtained as classification-by-detection.

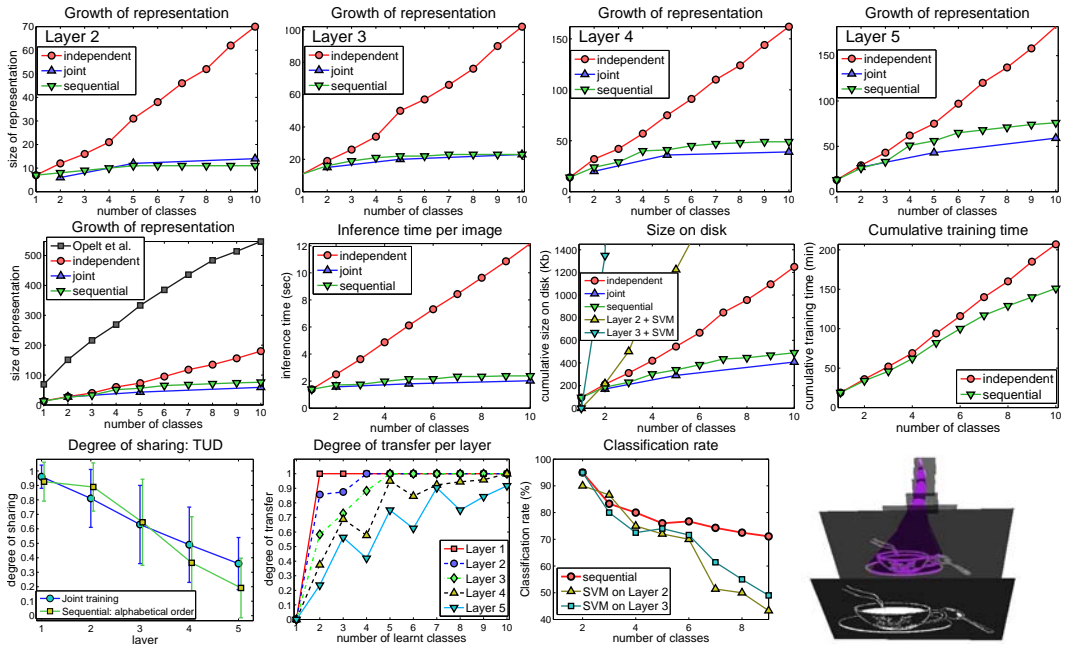

Figure 4: Results on TUD-10. Top: (1-4) repr. size as a function of the number of learned classes. Middle: 5.) repr. size compared to [4], 6.) size of hierarchy on disk, 7.) avg. inference time per image, 8.) cumulative train. time. Bottom: degree of 9.) sharing and 10.) transfer, 11.) classif. rates, 12.) example detection of cup.

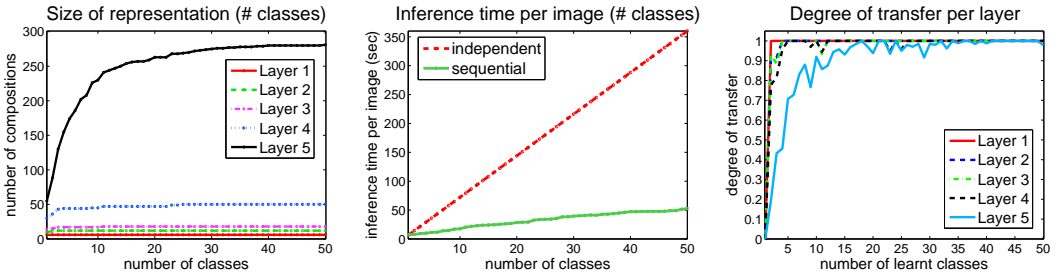

Figure 5: Results on 50 object classes from LabelMe [1]. From left to right: Size of representation (number of compositions per layer), inference times, and deg_transfer, all as a function of the number of learned classes.

Table 1: Dataset information.

| dataset | ETH shape | | | | | TUD shape1 (train) + TUD shape2 (test) | | | | | | | | | | Graz | | UIUC | Weizm. |
|---|---|---|---|---|---|---|---|---|---|---|---|---|---|---|---|---|---|---|---|
| class | apple | bottle | giraffe | mug | swan | cup | fork | hammer | knife | mug | pan | pliers | pot | saucepan | scissors | cow | person | car | horse |
| # train | 19 | 21 | 30 | 24 | 15 | 20 | 20 | 20 | 20 | 20 | 20 | 20 | 20 | 20 | 20 | 20 | 19 | 40 | 20 |
| # test | 21 | 27 | 57 | 24 | 17 | 10 | 10 | 10 | 10 | 10 | 10 | 10 | 10 | 10 | 10 | 65 | 19 | 108 | 228 |

Table 2: Comparison of detection rates with related work. Left: Average detection-rate (in %) at 0.4 FPPI for the related work, while we also report the actual FPPI, for the ETH shape database. Right: Recall at EER for various classes. The approaches that use more than just contour information (and are thus not directly comparable to ours) are shaded gray.

| | applelogo | bottle | giraffe | mug | swan | average |
|---|---|---|---|---|---|---|
| [24] | 83.2(1.7) | 83.3(7.5) | 58.6(14.6) | 83.6(8.6) | 75.4(13.4) | 76.8 |
| [26] | 89.9(4.5) | 76.8(6.1) | 90.5(5.4) | 82.7(5.1) | 84.0(8.4) | 84.8 |
| ind. train. | 87.3(2.6) | **86.2(2.8)** | 83.3(4.3) | **84.6(2.3)** | 78.2(5.4) | 83.7 |
| | 0.32 FPPI | 0.36 FPPI | 0.21 FPPI | 0.27 FPPI | 0.26 FPPI | |

| | cow | horse | car.scale | person |
|---|---|---|---|---|
| Related work | 100.[4] | 89.0[23] | 90.6[27] | 52.6[4] |
| | 100.[7] | 93.0[28] | 93.5[29] | 52.4[23] |
| | | / | 92.1[13] | 50.0[28] |
| ind. train. | 98.5 | **94.3** | **93.5** | **60.4** |

Table 3: Results for different learning types (number of compositions per layer) on the cow-horse, and car-person classes.

| class | size of representation | | | | | | | | | | | | | | | | train. time (min) | | | | infer. time (sec) | | | | rec. at EER (%) | | | |
|---|---|---|---|---|---|---|---|---|---|---|---|---|---|---|---|---|---|---|---|---|---|---|---|---|---|---|---|---|
| | $I$ | | | | $J$ | | | | $S_1$ (1+2) | | | | $S_2$ (2+1) | | | | $I$ | $J$ | $S_1$ | $S_2$ | $I$ | $J$ | $S_1$ | $S_2$ | $I$ | $J$ | $S_1$ | $S_2$ |
| | L2 | L3 | L4 | L5 | L2 | L3 | L4 | L5 | L2 | L3 | L4 | L5 | L2 | L3 | L4 | L5 | | | | | | | | | | | | |
| cow (1) | 17 | 17 | 23 | 25 | 17 | 25 | 27 | 25 | 17 | 17 | 23 | 25 | 14 | 17 | 20 | 20 | 25 | / | 25 | 19 | 1.9 | 2.0 | 2.1 | 2.0 | 96.9 | 96.9 | 96.9 | 98.5 |
| horse (2) | 12 | 12 | 18 | 24 | 12 | 26 | 26 | 27 | 18 | 18 | 24 | 27 | 12 | 12 | 18 | 24 | 20 | / | 17 | 20 | 2.3 | 2.6 | 2.7 | 2.7 | 94.3 | 93.4 | 93.4 | 94.3 |
| cow+hrs. | 29 | 29 | 41 | 49 | 18 | 26 | 30 | 36 | 21 | 29 | 33 | 38 | 19 | 29 | 29 | 38 | 45 | 65 | 42 | 39 | 4.3 | 2.5 | 2.6 | 2.5 | 95.6 | 95.6 | 95.6 | **96.4** |
| car (1) | 6 | 10 | 13 | 16 | 16 | 16 | 20 | 20 | 6 | 10 | 13 | 16 | 11 | 13 | 18 | 18 | 35 | / | 35 | 33 | 3.4 | 5.2 | 5.3 | 5.4 | 93.5 | 91.7 | 93.5 | 92.4 |
| person (2) | 9 | 16 | 19 | 21 | 11 | 12 | 14 | 22 | 11 | 12 | 16 | 23 | 9 | 16 | 19 | 21 | 17 | / | 15 | 17 | 2.3 | 2.6 | 2.8 | 3.0 | 60.4 | 58.3 | 56.3 | 60.4 |
| car+prsn. | 15 | 26 | 32 | 37 | 18 | 25 | 30 | 42 | 12 | 19 | 27 | 42 | 11 | 25 | 31 | 38 | 52 | 85 | 50 | 50 | 6.3 | 4.8 | 4.9 | 5.0 | **77.0** | 75.0 | 74.9 | 76.4 |

Table 4: Results for different learning types on the 5−class ETH shape dataset.

| class | size of representation | | | | | | | | | | | | trn.time(min) | | | infr.time(sec) | | | det. rate (%) | | | FPPI | | |
|---|---|---|---|---|---|---|---|---|---|---|---|---|---|---|---|---|---|---|---|---|---|---|---|---|
| | $I$ | | | | $J$ | | | | $S$, mean (std) - over 3 runs | | | | $I$ | $J$ | $S$ | $I$ | $J$ | $S$ | $I$ | $J$ | $S$ | $I$ | $J$ | $S$ |
| | L2 | L3 | L4 | L5 | L2 | L3 | L4 | L5 | L2 | L3 | L4 | L5 | | | | | | | | | | | | |
| applelogo | 11 | 30 | 27 | 28 | 15 | 21 | 21 | 28 | 10(0.6) | 25(1.7) | 27(12.7) | 23(7.5) | 23 | / | 23 | 3.6 | 11.1 | 12.1 | 88.6 | 86.4 | 86.4 | 0.34 | 0.27 | 0.28 |
| bottle | 7 | 11 | 22 | 22 | 16 | 16 | 21 | 22 | 9(0.6) | 22(8.1) | 28(2) | 22(3.6) | 25 | / | 21 | 3.4 | 11.1 | 12.1 | 85.5 | 80.0 | 80.0 | 0.4 | 0.34 | 0.32 |
| giraffe | 5 | 13 | 22 | 37 | 19 | 19 | 26 | 26 | 10(0.6) | 28(5.9) | 35(1.7) | 30(1.2) | 31 | / | 26 | 3.2 | 11.1 | 12.1 | 82.4 | 81.3 | 84.6 | 0.19 | 0.16 | 0.18 |
| mug | 9 | 16 | 25 | 23 | 19 | 19 | 25 | 34 | 10(0.6) | 25(7.9) | 30(4.7) | 29(4.9) | 31 | / | 18 | 3.6 | 11.1 | 12.1 | 84.9 | 83.3 | 83.3 | 0.31 | 0.22 | 0.22 |
| swan | 11 | 18 | 29 | 26 | 20 | 20 | 26 | 27 | 10(0) | 23(6.4) | 30(4.0) | 27(1.5) | 17 | / | 12 | 2.8 | 11.1 | 12.1 | 75.8 | 69.7 | 72.7 | 0.28 | 0.22 | 0.21 |
| **all** | 43 | 88 | 125 | 136 | 22 | 32 | 32 | 55 | 11(0.6) | 33(2.5) | 61(9.5) | 79(13.7) | 127 | 235 | 100 | 16.6 | 11.1 | 12.1 | **83.4** | 80.1 | 81.4 | 0.30 | **0.24** | **0.24** |

## Footnotes

[1]The number of layers depends on the objects' size in the training images (it is logarithmic with the number of non-overlapping contour fragments in an image). To enable a consistent evaluation of feature sharing, etc, we have scaled the training images in a way which produced the whole-shape models at layer 6 for each class.

# References

[1] Russell, B., Torralba, A., Murphy, K., and Freeman, W. T. (2008) Labelme: a database and web-based tool for image annotation. *IJCV*, **77**, 157–173.

[2] Leibe, B., Leonardis, A., and Schiele, B. (2008) Robust object detection with interleaved categorization and segmentation. *IJCV*, **77**, 259–289.

[3] Torralba, A., Murphy, K. P., and Freeman, W. T. (2007) Sharing visual features for multiclass and multi-view object detection. *IEEE PAMI*, **29**, 854–869.

[4] Opelt, A., Pinz, A., and Zisserman, A. (2008) Learning an alphabet of shape and appearance for multi-class object detection. *IJCV*, **80**, 16–44.

[5] Fei-Fei, L., Fergus, R., and Perona, P. (2004) Learning generative visual models from few training examples: an incremental bayesian approach tested on 101 object categories. *IEEE CVPR'04 Workshop on Generative-Model Based Vision*.

[6] Krempp, S., Geman, D., and Amit, Y. (2002) Sequential learning of reusable parts for object detection. Tech. rep.

[7] Todorovic, S. and Ahuja, N. (2007) Unsupervised category modeling, recognition, and segmentation in images. *IEEE PAMI*.

[8] Zhu, S. and Mumford, D. (2006) A stochastic grammar of images. *Found. and Trends in Comp. Graphics and Vision*, **2**, 259–362.

[9] Ranzato, M. A., Huang, F.-J., Boureau, Y.-L., and LeCun, Y. (2007) Unsupervised learning of invariant feature hierarchies with applications to object recognition. *CVPR*.

[10] Ullman, S. and Epshtein, B. (2006) *Visual Classification by a Hierarchy of Extended Features.*. Towards Category-Level Object Recognition, Springer-Verlag.

[11] Sivic, J., Russell, B. C., Zisserman, A., Freeman, W. T., and Efros, A. A. (2008) Unsupervised discovery of visual object class hierarchies. *CVPR*.

[12] Bart, I., Porteous, E., Perona, P., and Wellings, M. (2008) Unsupervised learning of visual taxonomies. *CVPR*.

[13] Fidler, S. and Leonardis, A. (2007) Towards scalable representations of visual categories: Learning a hierarchy of parts. *CVPR*.

[14] Scalzo, F. and Piater, J. H. (2005) Statistical learning of visual feature hierarchies. *W. on Learning, CVPR*.

[15] Zhu, L., Lin, C., Huang, H., Chen, Y., and Yuille, A. (2008) Unsupervised structure learning: Hierarchical recursive composition, suspicious coincidence and competitive exclusion. *ECCV*, vol. 2, pp. 759–773.

[16] Fleuret, F. and Geman, D. (2001) Coarse-to-fine face detection. *IJCV*, **41**, 85–107.

[17] Schwartz, J. and Felzenszwalb, P. (2007) Hierarchical matching of deformable shapes. *CVPR*.

[18] Ommer, B. and Buhmann, J. M. (2007) Learning the compositional nature of visual objects. *CVPR*.

[19] Serre, T., Wolf, L., Bileschi, S., Riesenhuber, M., and Poggio, T. (2007) Object recognition with cortex-like mechanisms. *IEEE PAMI*, **29**, 411–426.

[20] Sudderth, E., Torralba, A., Freeman, W. T., and Willsky, A. (2008) Describing visual scenes using transformed objects and parts. *IJCV*, pp. 291–330.

[21] Fidler, S., Boben, M., and Leonardis, A. (2009) Optimization framework for learning a hierarchical shape vocabulary for object class detection. *BMVC*.

[22] Agarwal, S., Awan, A., and Roth, D. (2004) Learning to detect objects in images via a sparse, part-based representation. *IEEE PAMI*, **26**, 1475–1490.

[23] Shotton, J., Blake, A., and Cipolla, R. (2008) Multi-scale categorical object recognition using contour fragments. *PAMI*, **30**, 1270–1281.

[24] Ferrari, V., Fevrier, L., Jurie, F., and Schmid, C. (2007) Accurate object detection with deformable shape models learnt from images. *CVPR*.

[25] Stark, M. and Schiele, B. (2007) How good are local features for classes of geometric objects? *ICCV*.

[26] Fritz, M. and Schiele, B. (2008) Decomposition, discovery and detection of visual categories using topic models. *CVPR*.

[27] Mutch, J. and Lowe, D. G. (2006) Multiclass object recognition with sparse, localized features. *CVPR*, pp. 11–18.

[28] Shotton, J., Blake, A., and Cipolla, R. (2008) Efficiently combining contour and texture cues for object recognition. *BMVC*.

[29] Ahuja, N. and Todorovic, S. (2008) Connected segmentation tree – a joint representation of region layout and hierarchy. *CVPR*.

